# The Lovász $\vartheta$ function, SVMs and finding large dense subgraphs

**Vinay Jethava** *
Computer Science & Engineering Department,
Chalmers University of Technology
412 96, Goteborg, SWEDEN
jethava@chalmers.se

**Anders Martinsson**
Department of Mathematics,
Chalmers University of Technology
412 96, Goteborg, SWEDEN
andemar@student.chalmers.se

**Chiranjib Bhattacharyya**
Department of CSA,
Indian Institute of Science
Bangalore, 560012, INDIA
chiru@csa.iisc.ernet.in

**Devdatt Dubhashi**
Computer Science & Engineering Department,
Chalmers University of Technology
412 96, Goteborg, SWEDEN
dubhashi@chalmers.se

## Abstract

The Lovász $\vartheta$ function of a graph, a fundamental tool in combinatorial optimization and approximation algorithms, is computed by solving a SDP. In this paper we establish that the Lovász $\vartheta$ function is equivalent to a kernel learning problem related to one class SVM. This interesting connection opens up many opportunities bridging graph theoretic algorithms and machine learning. We show that there exist graphs, which we call $\mathbf{SVM} - \vartheta$ graphs, on which the Lovász $\vartheta$ function can be approximated well by a one-class SVM. This leads to novel use of SVM techniques for solving algorithmic problems in large graphs e.g. identifying a planted clique of size $\Theta(\sqrt{n})$ in a random graph $G(n, \frac{1}{2})$. A classic approach for this problem involves computing the $\vartheta$ function, however it is not scalable due to SDP computation. We show that the random graph with a planted clique is an example of $\mathbf{SVM} - \vartheta$ graph. As a consequence a SVM based approach easily identifies the clique in large graphs and is competitive with the state-of-the-art. We introduce the notion of *common orthogonal labelling* and show that it can be computed by solving a Multiple Kernel learning problem. It is further shown that such a labelling is extremely useful in *identifying a large common dense subgraph in multiple graphs*, which is known to be a computationally difficult problem. The proposed algorithm achieves an order of magnitude scalability compared to state of the art methods.

## 1 Introduction

The Lovász $\vartheta$ function [19] plays a fundamental role in modern combinatorial optimization and in various approximation algorithms on graphs, indeed Goemans was led to say It seems all roads lead to $\vartheta$ [10]. The function is an instance of semidefinite programming(SDP) and hence computing it is an extremely demanding task even for moderately sized graphs. In this paper we establish that the $\vartheta$ function is equivalent to solving a kernel learning problem in the one-class SVM setting. This surprising connection opens up many opportunities which can benefit both graph theory and machine learning. In this paper we exploit this novel connection to show an interesting application of the SVM setup for identfying large dense subgraphs. More specifically we make the following contributions.

## 1.1 Contributions:

**1.**We give a new SDP characterization of Lovász $\vartheta$ function,

$$\min_{\mathbf{K} \in \mathcal{K}(G)} \omega(\mathbf{K}) = \vartheta(G)$$

where $\omega(\mathbf{K})$ is computed by solving an one-class SVM. The matrix $\mathbf{K}$ is a kernel matrix, associated with any orthogonal labelling of $G$. This is discussed in Section 2.

**2.** Using an easy to compute orthogonal labelling we show that there exist graphs, which we call $\mathbf{SVM} - \vartheta$ graphs, on which Lovász $\vartheta$ function can be well approximated by solving an one-class SVM. This is discussed in Section 3.

**3.** The problem of finding a *large common dense subgraph in multiple graphs* arises in a variety of domains including Biology, Internet, Social Sciences [18]. Existing state-of-the-art methods [14] are enumerative in nature and has complexity exponential in the size of the subgraph. We introduce the notion of *common orthogonal labelling* which can be used to develop a formulation which is close in spirit to a Multiple Kernel Learning based formulation. Our results on the well known DIMACS benchmark dataset show that it can identify large common dense subgraphs in wide variety of settings, beyond the realm of state-of-the-art methods. This is discussed in Section 4.

**4.** Lastly, in Section 5, we show that the famous planted clique problem, can be easily solved for large graphs by solving an one-class SVM. Many problems of interest in the area of machine learning can be reduced to the problem of detecting planted clique, e.g detecting correlations [1, section 4.6], correlation clustering [21] etc. The planted clique problem consists of identifying a large clique in a random graph. There is an elegant approach for identifying the planted clique by computing the Lovász $\vartheta$ function [8], however it is not practical for large graphs as it requires solving an SDP. We show that the graph associated with the planted clique problem is a $\mathbf{SVM} - \vartheta$ graph, paving the way for identifying the clique by solving an one-class SVM. Apart from the method based on computing the $\vartheta$ function, there are other methods for planted clique identification, which do not require solving an SDP [2, 7, 24]. Our result is also competitive with the state-of-the-art non-SDP based approaches [24].

**Notation** We denote the Euclidean norm by $\| \cdot \|$ and the infinity norm by $\| \cdot \|_\infty$. Let $\mathcal{S}^{d-1} = \{\mathbf{u} \in \mathbb{R}^d | \|\mathbf{u}\| = 1\}$ denote a $d$ dimensional sphere. Let $\mathbf{S}_n$ denote the set of $n \times n$ square symmetric matrices and $\mathbf{S}_n^+$ denote $n \times n$ square symmetric positive semidefinite matrices. For any $\mathbf{A} \in \mathbf{S}_n$ we denote the eigenvalues $\lambda_1(\mathbf{A}) \geq \ldots \geq \lambda_n(\mathbf{A})$. $\texttt{diag(r)}$ will denote a diagonal matrix with diagonal entries defined by components of $r$. We denote the one-class SVM objective function by

$$\omega(\mathbf{K}) = \max_{\alpha_i \geq 0, i=1,\ldots,n} \underbrace{\left(2 \sum_{i=1}^n \alpha_i - \sum_{i=1}^n \alpha_i \alpha_j K_{ij}\right)}_{f(\alpha; \mathbf{K})} \tag{1}$$

where $\mathbf{K} \in \mathbf{S}_n^+$. Let $G = (V, E)$ be a graph on vertices $V = \{1, \ldots, n\}$ and edge set $E$. Let $\mathbf{A} \in \mathbf{S}^n$ denote the adjacency matrix of $G$ where $A_{ij} = 1$ if edge $(i, j) \in E$, and 0 otherwise. An eigenvalue of graph $G$ would mean the eigenvalue of the adjacency matrix of $G$. Let $\bar{G}$ denote the complement graph of $G$. The adjacency matrix of $\bar{G}$ is $\bar{\mathbf{A}} = \mathbf{e}\mathbf{e}^\top - \mathbf{I} - \mathbf{A}$, where $\mathbf{e} = [1, 1, \ldots, 1]^\top$ is a vector of length $n$ containing all 1's, and $\mathbf{I}$ denotes the identity matrix. Let $G_S = (S, E_S)$ denote the subgraph induced by $S \subseteq V$ in graph $G$; having density $\gamma(G_S)$ is given by $\gamma(G_S) = |E_S|/\binom{|S|}{2}$. Let $N_i(G) = \{j \in V : (i, j) \in E\}$ denote the set of neighbours of vertex $i$ in graph $G$, and degree of node $i$ to be $d_i(G) = |N_i(G)|$. An independent set in $G$ (a clique in $\bar{G}$ is a subset of vertices $S \subseteq V$ for which no (every) pair of vertices has an edge in $G$ (in $\bar{G}$). The notation is standard e.g. see [3].

## 2 Lovász $\vartheta$ function and Kernel learning

Consider the problem of embedding a graph $G = (V, E)$ on a $d$ dimensional unit sphere $S^{d-1}$. The study of this problem was initiated in [19] which introduced the idea of *orthogonal labelling*: An

orthogonal labelling of graph $G = (V, E)$ with $|V| = n$, is a matrix $U = [\mathbf{u}_1, \ldots, \mathbf{u}_n] \in \mathbb{R}^{d \times n}$ such that $\mathbf{u}_i^\top \mathbf{u}_j = 0$ whenever $(i, j) \notin E$ and $\mathbf{u}_i \in \mathcal{S}^{d-1} \; \forall \, i = 1, \ldots, n$.

An orthogonal labelling defines an embedding of a graph on a $d$ dimensional unit sphere: for every vertex $i$ there is a vector $\mathbf{u}_i$ on the unit sphere and for every $(i, j) \notin E$ $\mathbf{u}_i$ and $\mathbf{u}_j$ are orthogonal. Using the notion of orthogonal labellings, [19] defined a function, famously known as Lovász $\vartheta$ function, which upper bounds the size of maximum independent set. More specifically

$$\text{for any graph } G \; : \; \mathbf{ALPHA}(G) \leq \vartheta(G),$$

where $\mathbf{ALPHA}(G)$ is the size of the largest independent set. Finding large independent sets is a fundamental problem in algorithm design and analysis and computing $\mathbf{ALPHA}(G)$ is a classic NP-hard problem which is even very hard even to approximate [11]. However, the Lovász function $\vartheta(G)$ gives a tractable upper-bound and since then Lovász $\vartheta$ function has been extensively used in solving a variety of algorithmic problems e.g. [6]. It maybe useful to recall the definition of Lovász $\vartheta$ function. Denote the set of all possible orthogonal labellings of $G$ by $Lab(G) = \{\mathbf{U} = [\mathbf{u}_1, \ldots, \mathbf{u}_n] | \mathbf{u}_i \in \mathcal{S}^{d-1}, \; \mathbf{u}_i^\top \mathbf{u}_j = 0 \, \forall (i, j) \notin E\}$.

$$\vartheta(G) = \min_{\mathbf{U} \in Lab(G)} \min_{\mathbf{c} \in \mathcal{S}^{d-1}} \max_i \frac{1}{(\mathbf{c}^\top \mathbf{u}_i)^2} \tag{2}$$

There exist several other equivalent definitions of $\vartheta$, for a comprehensive discussion see [16].

However computation of Lovász $\vartheta$ function is not practical even for moderately sized graphs as it requires solving a semidefinite program on a matrix which is of the size of the graph. In the following theorem, we show that there exist connections between the $\vartheta$ function and the SVM formulation.

**Theorem 2.1.** *For a undirected graph $G = (V, E)$, with $|V| = n$, let $\mathcal{K}(G) := \{\mathbf{K} \in \mathbf{S}_n^+ \mid K_{ii} = 1, i \in [n], K_{ij} = 0, (i, j) \notin E\}$ Then, $\vartheta(G) = \min_{\mathbf{K} \in \mathcal{K}(G)} \omega(\mathbf{K})$*

*Proof.* We begin by noting that any $\mathbf{K} \in \mathcal{K}(G)$ is positive semidefinite and hence there exists $\mathbf{U} \in \mathbb{R}^{d \times n}$ such that $\mathbf{K} = \mathbf{U}^\top \mathbf{U}$. Note that $K_{ij} = \mathbf{u}_i^\top \mathbf{u}_j$ where $\mathbf{u}_i$ is a column of $\mathbf{U}$. Hence by inspection it is clear that the columns of $\mathbf{U}$ defines an orthogonal labelling on $G$, i.e $\mathbf{U} \in Lab(G)$. Using a similar argument we can show that for any $\mathbf{U} \in Lab(G)$, the matrix $\mathbf{K} = \mathbf{U}^\top \mathbf{U}$, is an element of $\mathcal{K}(G)$. The set of valid kernel matrices $\mathcal{K}(G)$ is thus equivalent to $Lab(G)$. Note that if $\mathbf{U}$ is a labelling then $\mathbf{U} = \mathbf{U}\texttt{diag}(\epsilon)$ is also an orthogonal labelling for any $\epsilon^\top = [\epsilon_1, \ldots, \epsilon_n]$, $\epsilon_i = \pm 1 \; i = 1, \ldots, n$. It thus suffices to consider only those labellings for which $\mathbf{c}^\top \mathbf{u}_i \geq 0 \; \forall i = 1, \ldots, n$ holds. For a fixed $\mathbf{c}$ one can write $\max_i \frac{1}{(\mathbf{c}^\top \mathbf{u}_i)^2} = \min_t t^2$ subject to $\frac{1}{\mathbf{c}^\top \mathbf{u}_i} \leq t$. This is true because the minimum over $t$ is attained at $\max_i \frac{1}{\mathbf{c}^\top \mathbf{u}_i}$. Setting $\mathbf{w} = 2t\mathbf{c}$ yields the following relation $\min_{\mathbf{c} \in \mathcal{S}^{d-1}} \max_i \frac{1}{(\mathbf{c}^\top \mathbf{u}_i)^2} = \min_{\mathbf{w} \in \mathbb{R}^d} \frac{\|\mathbf{w}\|^2}{4}$ with constraints $\mathbf{w}^\top \mathbf{u}_i \geq 2$. This establishes that for a labelling, $\mathbf{U}$, the optimal $\mathbf{c}$ is obtained by solving an one-class SVM. Application of strong duality immediately leads to the claim $\min_{\mathbf{c} \in \mathcal{S}^{d-1}} \max_i \frac{1}{(\mathbf{c}^\top \mathbf{u}_i)^2} = \omega(\mathbf{K})$ where $\mathbf{K} = \mathbf{U}^\top \mathbf{U}$ and $\omega(\mathbf{K})$ is defined in (1). As there is a correspondence between each element of $Lab(G)$ and $\mathcal{K}$ minimization of $\omega(\mathbf{K})$ over $\mathcal{K}$ is equivalent to computing the $\vartheta(G)$ function. $\qquad\square$

This is a significant result which establishes connections between two well studied formulations, namely $\vartheta$ function and the SVM formulation. An important consequence of Theorem 2.1 is an easily computable upperbound on $\vartheta(G)$ namely that for any graph $G$

$$\mathbf{ALPHA}(G) \leq \vartheta(G) \leq \omega(\mathbf{K}) \; \forall \mathbf{K} \in \mathcal{K}(G) \tag{3}$$

Since solving $\omega(\mathbf{K})$ is a convex quadratic program, it is indeed a computationally efficient alternative to the $\vartheta$ function. In fact we will show that there exist families of graphs for which $\vartheta(G)$ can be approximated to within a constant factor by $\omega(\mathbf{K})$ for suitable $\mathbf{K}$. Theorem 2.1 is closely related to the following result proved in [20].

**Theorem 2.2.** *[20] For a graph $G = (V, E)$ with $|V| = n$ let $\mathbf{C} \in \mathbf{S}_n$ matrix with $C_{ij} = 0$ whenever $(i, j) \notin E$. Then,*

$$\vartheta(G) = min_{\mathbf{C}} \quad v(G, \mathbf{C}) \left( = \max_{\mathbf{x} \geq 0} 2\mathbf{x}^\top e - \mathbf{x}^\top \left( \frac{\mathbf{C}}{-\lambda_n(\mathbf{C})} + \mathbf{I} \right) \mathbf{x} \right)$$

*Proof.* See [20] □

See that for any feasible $\mathbf{C}$ the matrix $\mathbf{I} + \frac{\mathbf{C}}{-\lambda_n(\mathbf{C})} \in \mathcal{K}(G)$. Theorem 2.1 is a restatement of Theorem 2.2, but has the additional advantage that the stated optimization problem can be solved as an SDP. The optimization problem $min_{\mathbf{C}} v(G, \mathbf{C})$ with constraints on $\mathbf{C}$ is not an SDP. If we fix $\mathbf{C} = \mathbf{A}$, the adjacency matrix, we obtain a very interesting orthogonal labelling, which we will refer to as **LS** labelling, introduced in [20]. Indeed there exists family of graphs, called $Q$ graphs for which **LS** labelling yields the interesting result $\mathbf{ALPHA}(G) = v(G, \mathbf{A})$, see [20]. Indeed on a $Q$ graph one does not need to compute a SDP, but can solve an one-class SVM, which has obvious computational benefits. Inspired by this result, in the remaining part of the paper, we study this labelling more closely. As a labelling is completely defined by the associated kernel matrix, we refer to the following kernel as the **LS** labelling,

$$\mathbf{K} = \frac{\mathbf{A}}{\rho} + \mathbf{I} \text{ where } \rho \geq -\lambda_n(\mathbf{A}). \tag{4}$$

## 3  SVM$-\vartheta$ graphs: Graphs where $\vartheta$ function can be approximated by SVM

We now introduce a class of graphs on which $\vartheta$ function can be well approximated by $\omega(\mathbf{K})$ for $\mathbf{K}$ defined by (4). In the spirit of approximation algorithms we define:

**Definition 3.1.** *A graph $G$ is a* **SVM**$-\vartheta$ *graph if $\omega(\mathbf{K}) \leq (1 + O(1))\vartheta(G)$ where $\mathbf{K}$ is a* **LS** *labelling.*

Such classes of graphs are interesting because on them, one can approximate the Lovász $\vartheta$ function by solving an SVM, instead of an SDP, which in turn can be extremely useful in the design and analysis of approximation algorithms. We will demosntrate two examples of **SVM**$-\vartheta$ graphs namely (a.) the Erdös–Renyi random graph $G(n, 1/2)$ and (b.) a planted variation. Here the relaxation $\omega(\mathbf{K})$ could be used in place of $\vartheta(G)$, resulting in algorithms with the same quality guarantees but with faster running time – in particular, this will allow the algorithms to be scaled to large graphs.

The classical Erdös-Renyi random graph $G(n, 1/2)$ has $n$ vertices and each edge $(i, j)$ is present independently with probability $1/2$. We list a few facts about $G(n, 1/2)$ that will be used repeatedly.

**Fact 3.1.** *For $G(n, 1/2)$,*

- *With probability $1 - O(1/n)$, the degree of each vertex is in the range $n/2 \pm O(\sqrt{n \log n})$.*

- *With probability $1 - e^{-n^c}$ for some $c > 0$, the maximum eigenvalue is $n/2 \pm o(n)$ and the minimum eigenvalue is in the range $[-\sqrt{n}, \sqrt{n}]$ [9].*

**Theorem 3.1.** *Let $\epsilon > \sqrt{2} - 1$. For $G = G(n, 1/2)$, with probability $1 - O(1/n)$, $\omega(\mathbf{K}) \leq (1 + \epsilon)\vartheta(G)$ where $\mathbf{K}$ is defined in (4) with $\rho = \frac{1+\epsilon}{\sqrt{2}}\sqrt{n}$.*

*Proof.* We begin by considering the case for $\rho = (1 + \frac{\delta}{2})\sqrt{n}$. By Fact 3.1 for all choices of $\delta > 0$, the minimum eigenvalue of $\frac{1}{\rho}\mathbf{A} + \mathbf{I}$ is, almost surely, greater than 0 which implies that $f(\alpha, \mathbf{K})$ (see (1)) is strongly concave. For such functions KKT conditions are neccessary and sufficient for optimality. The KKT conditions for a $G(n, \frac{1}{2})$ are given by the following equation

$$\alpha_i + \frac{1}{\rho} \sum_{(i,j) \in E} A_{i,j} \alpha_j = 1 + \mu_i, \ \mu_i \alpha_i = 0, \mu_i \geq 0 \tag{5}$$

As $\mathbf{A}$ is random we begin by analyzing the case for expectation of $\mathbf{A}$. Let $\mathbb{E}(\mathbf{A}) = \frac{1}{2}(\mathbf{e}\mathbf{e}^\top - I)$, be the expectation of $A$. For the given choice of $\rho$, the matrix $\tilde{\mathbf{K}} = \frac{\mathbb{E}(\mathbf{A})}{\rho} + \mathbf{I}$ is positive definite. More importantly $f(\alpha, \tilde{\mathbf{K}})$ is again strongly concave and attains maximum at a KKT point. By direct verification $\hat{\alpha} = \hat{\beta}\mathbf{e}$ where $\hat{\beta} = \frac{2\rho}{n-1+2\rho}$ satisfies

$$\alpha + \frac{1}{\rho}\mathbb{E}(\mathbf{A})\alpha = \mathbf{e}. \tag{6}$$

Thus $\hat\alpha$ is the KKT point for the problem,

$$\bar f = \max_{\alpha \geq 0} f(\alpha, \tilde{\mathbf{K}}) = \sum_{i=1}^{n} \hat\alpha - \hat\alpha^\top \left( \frac{\mathbb{E}(\mathbf{A})}{\rho} + \mathbf{I} \right) \hat\alpha = n\hat\beta \tag{7}$$

with the optimal objective function value $\bar f$. By choice of $\rho = (1 + \frac{\delta}{2})\sqrt{n}$ we can write $\hat\beta = 2\rho/n + O(1/n)$. Using the fact about degrees of vertices in $G(n, 1/2)$, we know that

$$\mathbf{a}_i^\top \mathbf{e} = \frac{n-1}{2} + \Delta_i \text{ with } |\Delta_i| \leq \sqrt{n \log n} \tag{8}$$

where $\mathbf{a}_i^\top$ is the $i$th row of the adjacency matrix $\mathbf{A}$. As a consequence we note that

$$\left| \hat\alpha_i + \frac{1}{\rho} \sum_j A_{ij} \hat\alpha_j - 1 \right| = \frac{\hat\beta}{\rho} \Delta_i \tag{9}$$

Recalling the definition of $f$ and using the above equation along with (8) gives

$$|f(\hat\alpha; \mathbf{K}) - \bar f| \leq n \frac{\hat\beta^2}{\rho} \sqrt{n \log n} \tag{10}$$

As noted before the function $f(\alpha; \mathbf{K})$ is strongly concave with $\nabla_\alpha^2 f(\alpha; \mathbf{K}) \preceq -\frac{\delta}{2+\delta}\mathbf{I}$ for all feasible $\alpha$. Recalling a useful result from convex optimization, see Lemma 3.1, we obtain

$$\omega(\mathbf{K}) - f(\hat\alpha; \mathbf{K}) \leq \left( 1 + \frac{1}{\delta} \right) \|\nabla f(\hat\alpha; \mathbf{K})\|^2 \tag{11}$$

Observing that $\nabla f(\alpha; \mathbf{K}) = 2(\mathbf{e} - \alpha - \frac{A}{\rho}\alpha)$ and using the relation between $\|\cdot\|_\infty$ and 2 norm along with (9) and (8) gives $\|\nabla f(\hat\alpha; \mathbf{K})\| \leq \sqrt{n}\|\nabla f(\hat\alpha; \mathbf{K})\|_\infty = 2n\frac{\hat\beta}{\rho}\sqrt{\log n}$. Plugging this estimate in (11) and using equation (10) we obtain $\omega(\mathbf{K}) \leq \hat f + O(\log n) = (2+\delta)\sqrt{n} + O(\log n)$ The second equality follows by plugging in the value of $\hat\beta$ in (7). It is well known [6] that $\vartheta(G) = \sqrt{2}\sqrt{n}$ for $G(n, \frac{1}{2})$ with high probability. One concludes that $\omega(\mathbf{K}) \leq \frac{2+\delta}{\sqrt{2}}\vartheta(G) + o(\sqrt{n})$ and the theorem follows by choice of $\delta$. $\qquad\square$

**Discussion:** Theorem 3.1 establishes that instead of SDP one can solve an SVM to evaluate $\vartheta$ function on $G(n, 1/2)$. Although it is well known that $\mathbf{ALPHA}(G(n, 1/2)) = 2\log n$ whp, there is no known polynomial time algorithm for computing the maximum independent set. [6] gives an approximation algorithm that finds an independent set in $G(n, p)$ which runs in expected polynomial time, via a computation of $\vartheta(G(n, p))$, which also applies to $p = 1/2$. The $\vartheta$ function also serves as a guarantee of the approximation which other algorithms a simple Greedy algorithm cannot give. Theorem 3.1 allows us to obtain similar guarantees but without the computational overhead of solving an SDP. Apart from finding independent sets computing $\vartheta(G(n, p))$ is also used as a subroutine in *colorability* [6], and here again one can use the SVM based approach to approximate the $\vartheta$ function.

Similar arguments show also that other families of graphs such as the 11 families of pseudo–random graphs described in [17] are $\mathbf{SVM} - \vartheta$ graphs.

**Lemma 3.1.** *[4] A function $g : C \subset \mathbb{R}^d \to \mathbb{R}$ is said to be strongly concave over $S$ if there exists $t > 0$ such that $\nabla^2 g(\mathbf{x}) \preceq -t\mathbf{I} \ \forall \ \mathbf{x} \in C$. For such functions one can show that if $p^* = \max_{\mathbf{x} \in C} g(\mathbf{x}) < \infty$ then*

$$\forall \mathbf{x} \in C \ \ p^* - g(\mathbf{x}) \leq \frac{1}{2t}\|\nabla g(\mathbf{x})\|^2$$

## 4 Dense common subgraph detection

The problem of finding a large dense subgraph in multiple graphs has many applications [23, 22, 18]. We introduce the notion of *common orthogonal labelling*, and show that it is indeed possible to recover dense regions in large graphs by solving a MKL problem. This constitutes significant progress with respect to state of the art enumerative methods [14].

**Problem definition** Let $\mathcal{G} = \{G^{(1)}, \ldots, G^{(M)}\}$ be a set of simple, undirected graphs $G^{(m)} = (V, E^{(m)})$ defined on vertex set $V = \{1, \ldots, n\}$. Find a *common* subgraph which is *dense* in all the graphs.

Most algorithms which attempts the problem of finding a dense region are enumerative in nature and hence do not scale well to finding large cliques. [14], first studied a related problem of finding all possible *common* subgraphs for a given choice of parameters $\{\gamma^{(1)}, \ldots, \gamma^{(M)}\}$ which is atleast $\gamma_i$ dense in $G^{(i)}$. In the worst case, the algorithm performs depth first search over space of $\binom{n}{n_T}$ possible cliques of size $n_T$. This has $\Theta(\binom{n}{n_T})$ space and time complexity, which makes it impractical for moderately large $n_T$. For example, finding quasicliques of size 60 requires 8 hours (see Section 6).

In the remainder of this section, we focus on finding a large common *sparse subgraph* in a given collection of graphs; with the observation that this is equivalent to finding a large *common dense subgraph in the set of complement graphs*. To this end we introduce the following definition

**Definition 4.1.** *Given simple unweighted graphs, $G^{(m)} = (V, E^{(m)})$ $m = 1, \ldots, M$ on a common vertex set $V$ with $|V| = n$, the common orthogonal labelling on all the labellings is given by $\mathbf{u}_i \in \mathcal{S}^{d-1}$ such that $\mathbf{u}_i^\top \mathbf{u}_j = 0$ if $(i,j) \notin E^{(m)} \forall m = 1, \ldots, M\}$.*[1]

Following the arguments of Section 2 it is immediate that size of the *largest common independent set* is upper bounded by $\min_{\mathbf{K} \in L} \omega(\mathbf{K})$ where $L = \{\mathbf{K} \in \mathbf{S}_n^+ \; : \; \mathbf{K}_{ii} = 1 \forall i \in [n], \mathbf{K}_{ij} = 0 \text{ whenever } (i,j) \notin E^{(m)} \forall m = 1, \ldots, M\}$. We wish to exploit this fact in identifying large common sparse regions in general graphs. Unfortunately this problem is a SDP and will not scale well to large graphs. Taking cue from MKL literature we pose a restricted version of the problem namely

$$\min_{\mathbf{K} = \sum_{m=1}^M \delta_m \mathbf{K}^{(m)}, \, \delta_m \geq 0 \, \sum_{m=1}^M \delta_m = 1} \omega(\mathbf{K}) \tag{12}$$

where $\mathbf{K}^{(m)}$ is an orthogonal labelling of $G^{(m)}$. Direct verification shows that any feasible $\mathbf{K}$ is also a common orthogonal labelling. Using the fact that $\forall x \in \mathbb{R}^M \; \min_{p_m \geq 0, \sum_{m=1}^M p_m = 1} p^\top x = \min_m x_m = \max\{t | x_m \geq t \, \forall \, m = 1, \ldots, M\}$ one can recast the optimization problem in (12) as follows

$$\max_{t \in \mathbb{R}, \alpha_i \geq 0} t \text{ s.t. } f(\alpha; \mathbf{K}^{(m)}) \geq t \, \forall \, m = 1, \ldots, M \tag{13}$$

where $\mathbf{K}^{(m)}$ is the **LS** labelling for $G^{(m)}, \forall m = 1, \ldots, M$. The above optimization can be readily solved by state of the art MKL solvers. This result allows us to build a *parameter free* common sparse subgraph (CSS) algorithm shown in Figure 1 having following advantages: it provides a theoretical bound on subgraph density (Claim 4.1 below); and, it requires no parameters from the user beyond the set of graphs $G^{(1)}, \ldots, G^{(M)}$.

Let $\alpha^*$ be the optimal solution in (13); and $SV = \{i : \alpha_i^* > 0\}$ and $S_1 = \{i : \alpha_i^* = 1\}$ with cardinalities $n_{sv} = |SV|$ and $n_1 = |S_1|$ respectively. Let $\bar{\alpha}_{\min, S}^{(m)} = \min_{i \in S} \frac{\sum_{j \in N_i(G_S^{(m)})} \alpha_j^*}{d_i(G_S^{(m)})}$ denote the average of the support vector coefficients in the neighbourhood $N_i(G_S^{(m)})$ of vertex $i$ in induced subgraph $G_S^{(m)}$ having degree $d_i(G_S^{(m)}) = |N_i(G_S^{(m)})|$. We define

$$T^{(m)} = \left\{ i \in SV : d_i(G_{SV}^{(m)}) < \frac{(1-c)\rho^{(m)}}{\bar{\alpha}_{\min, SV}^{(m)}} \right\} \text{ where } c = \min_{i \in SV} \alpha_i^* \tag{14}$$

**Claim 4.1.** *Let $T \subseteq V$ be computed as in Algorithm 1. The subgraph $G_T^{(m)}$ induced by $T$, in graph $G^{(m)}$, has density at most $\gamma^{(m)}$ where $\gamma^{(m)} = \frac{(1-c)\rho^{(m)}}{\bar{\alpha}_{\min, SV}(n_T - 1)}$*

$\boxed{\begin{array}{l} \alpha^* = \text{Use MKL solvers to solve eqn. (13)} \\ T = \cap_m T^{(m)} \, \{\text{eqn. (14)}\} \\ \text{Return } T \end{array}}$

Figure 1: Algorithm for finding common sparse subgraph: $T = CSS(G^{(1)}, \ldots, G^{(M)})$

*Proof.* (Sketch) At optimality, $t = \sum_{i=1}^n \alpha_i^*$. This allows us to write $0 \leq \sum_{i \in S} \alpha_i^*(2 - \alpha_i^* - \sum_{j \neq i} K_{ij}^{(m)} \alpha_j^*) - t$ as $0 \leq \sum_{i \in T}(1 - c - \frac{d_i(G_T^{(m)})}{\rho^{(m)}} \bar{\alpha}_{\min, SV}^{(m)})$ Dividing by $\binom{n_T}{2}$ completes the proof. $\square$

# 5 Finding Planted Cliques in $G(n, 1/2)$ graphs

Finding large cliques or independent sets is a computationally difficult problem even in random graphs. While it is known that the size of the largest clique or independent set in $G(n, 1/2)$ is $2 \log n$ with high probability, there is no known efficient algorithm to find a clique of size significantly larger than $\log n$ - even a cryptographic application was suggested based on this (see the discussion and references in the introduction of [8]).

**Hidden planted clique**   A random graph $G(n, 1/2)$ is chosen first and then a clique of size $k$ is introduced in the first $1, \ldots, k$ vertices. The problem is to identify the clique.

[8] showed that if $k = \Omega(\sqrt{n})$, then the hidden clique can be discovered in polynomial time by computing the Lovász $\vartheta$ function. There are other approaches [2, 7, 24] which do not require computing the $\vartheta$ function.

We consider the (equivalent) complement model $\overline{G}(n, 1/2, k)$ where a independent set is planted on the set of $k$ vertices. We show that in the regime $k = \Omega(\sqrt{n})$, $\bar{G}(n, 1/2, k)$ is a $\mathbf{SVM} - \vartheta$ graph. We will further demonstrate that as a consequence one can identify the hidden independent set with high probability by solving an SVM. The following is the main result of the section.

**Theorem 5.1.** *For $G = \bar{G}(n, 1/2, k)$ and $k = 2t\sqrt{n}$ for large enough constant $t \geq 1$ with $\mathbf{K}$ as in (4) and $\rho = \sqrt{n} + k/2$,*

$$\omega(\mathbf{K}) = 2(t+1)\sqrt{n} + O(\log n) = \left(1 + \frac{1}{t} + o(1)\right) \vartheta(G)$$

*with probability at least $1 - O(1/n)$.*

*Proof.* The proof is analogous to that of Theorem 3.1. Note that $|\lambda_n(G)| \leq \sqrt{n} + k/2$. First we consider the expected case where all vertices outside the planted part $S$ are adjacent to $k/2$ vertices in $S$ and $(n - k)/2$ vertices outside $S$. and all verties in the planted part have degree $(n - k)/2$. We check that $\alpha_i = 2(t+1)/\sqrt{n}$ for $i \notin S$ and $\alpha_i = 2(t+1)^2/\sqrt{n}$ for $i \in S$ satisfy KKT conditions with an error of $O(1/\sqrt{n})$. Now apply Chernoff bounds to conclude that with high probability, all vertices in $S$ have degree $(n - k)/2 \pm \sqrt{(n - k) \log(n - k)}$ and those outside $S$ are adjacent to $k/2 \pm \sqrt{k \log k}$ vertices in $S$ and to $(n - k)/2 \pm \sqrt{(n - k) \log(n - k)}$ vertices ouside $S$. Now we check that the same solution satisfies KKT conditions of $\bar{G}(n, 1/2, k)$ with an error of $\epsilon = O\left(\sqrt{\frac{\log n}{n}}\right)$. Using the same arguments as in the proof of Theorem 3.1, we conclude that $\omega(\mathbf{K}) \leq 2(t+1)\sqrt{n} + O(\log n)$. Since $\vartheta(G) = 2t\sqrt{n}$ for this case [8], the result follows. $\qquad\square$

The above theorem suggests that the planted independent set can be recovered by taking the top $k$ values in the optimal solution. In the experimental section we will discuss the performance of this *recovery* algorithm. The runtime of this algorithm is one call to SVM solver, which is considerably cheaper than the SDP option. Indeed the algorithm due to [8], requires computation of $\vartheta$ function. The current best known algorithm for $\vartheta$ computation has an $O(n^5 \log n)$[5], run time complexity. In contrast the proposed approach needs to solve an SVM and hence scales well to large graphs. Our approach is competitive with the state of the art [24] as it gives the same high probability guarantees and have the same running time, $O(n^2)$. Here we have assumed that we are working with a SVM solver which has a time complexity of $O(n^2)$ [13].

# 6 Experimental evaluation

**Comparison with exhaustive approach [14]**   We generate synthetic $m = 3$ random graphs over $n$ vertices with average density $\delta = 0.2$, and having single (common) quasi-clique of size $k = 2\sqrt{n}$ with density $\gamma = 0.95$ in all the three graphs. This is similar to the synthetic graphs generated in the original paper [see 14, Section 6.1.2]. We note that both our MKL-based approach and exhaustive search in [14] recovers the quasi-clique. However, the time requirements are drastically different. All experiments were conducted on a computer with 16 GB RAM and Intel $X3470$ quad-core processor running at 2.93 GHz. Three values of $k$ namely $k = 50, 60$ and $k = 100$ were used. It is interesting to note that CROCHET [14] took **2 hours** and **9 hours** for $k = 50$ and $k = 60$ sized cliques and failed to find a clique of size of 100. The corresponding numbers for MKL are **47.5,54.8** and **137.6 seconds** respectively.

**Common dense subgraph detection**   We evaluate our algorithm for finding large dense regions on the DIMACS Challenge graphs [2] [15], which is a comprehensive benchmark for testing of clique finding and related algorithms. For the families of dense graphs (*brock*, *san*, *sanr*), we focus on finding large dense region in the complement of the original graphs.

We run Algorithm 1 using SimpleMkl[3] to find large common dense subgraph. In order to evaluate the performance of our algorithm, we compute $\bar{a} = \max_m a^{(m)}$ and $\underline{a} = \min_m a^{(m)}$ where $a^{(m)} = \gamma(G_T^{(m)})/\gamma(G^{(m)})$ is relative density of induced subgraph (compared to original graph density); and $n_T/N$ is relative size of induced subgraph compared to original graph size. We want a high value of $n_T/N$; while $\underline{a}$ should not be lower than 1. Table 1 shows evaluation of Algorithm 1 on DIMACS dataset. We note that our algorithm finds a large subgraph (large $n_T/N$) with higher density compared to original graph in all of DIMACS graph classes making it suitable for finding large dense regions in multiple graphs. In all cases the size of the subgraph, $n_T$ was more than 100. The MKL experiments reported in Table 1 took less than 1 minute (for each graph family); while the algorithm in [14] aborts after several hours due to memory constraints.

**Planted clique recovery**   We generate 100 random graphs based on planted clique model $G(n, 1/2, k)$ where $n = 30000$ and hidden clique size $k = 2t\sqrt{n}$ for each choice of $t$. We evaluate the recovery algorithm discussed in Section 4.2. The SVM problem is solved using Libsvm[4]. *For $t \geq 2$ we find perfect recovery of the clique on all the graphs*, which is agreement with Theorem 5.1.

It is worth noting that the approach takes 10 minutes to recover the clique in this graph of 30000 vertices which is far beyond the scope of SDP based procedures.

| Graph family | $N$ | $M$ | $\frac{n_T}{N}$ | $\bar{a}$ | $\underline{a}$ |
|---|---|---|---|---|---|
| c-fat200 | 200 | 3 | 0.50 | 2.12 | 0.99 |
| c-fat500 | 500 | 4 | 0.31 | 3.57 | 1.01 |
| brock200‡ | 200 | 4 | 0.41 | 1.36 | 0.99 |
| brock400‡ | 400 | 4 | 0.50 | 1.15 | 1.05 |
| brock800‡ | 800 | 4 | 0.50 | 1.08 | 1.01 |
| p_hat300 | 300 | 3 | 0.53 | 1.53 | 1.15 |
| p_hat500 | 500 | 3 | 0.48 | 1.55 | 1.17 |
| p_hat700 | 700 | 3 | 0.45 | 1.58 | 1.18 |
| p_hat1000 | 1000 | 3 | 0.43 | 1.60 | 1.19 |
| p_hat1500 | 1500 | 3 | 0.38 | 1.63 | 1.20 |
| san200‡ | 200 | 5 | 0.50 | 1.51 | 1.08 |
| san400‡ | 400 | 3 | 0.42 | 1.19 | 1.02 |
| sanr200‡ | 200 | 2 | 0.39 | 1.86 | 1.04 |
| sanr400‡ | 400 | 2 | 0.43 | 1.20 | 1.02 |

Table 1: Common dense subgraph recovery on multiple graphs in DIMACS dataset. Here $\bar{a}$ and $\underline{a}$ denote the maximum and minimum relative density of the induced subgraph (relative to density of the original graph) and $n_T/N$ is the relative size of the induced subgraph compared to original graph size.

## 7   Conclusion

In this paper we have established that the Lovász $\vartheta$ function, well studied in graph theory can be linked to the one-class SVM formulation. This link allows us to design scalable algorithms for computationally difficult problems. In particular we have demonstrated that finding a common dense region in multiple graphs can be solved by a MKL problem, while finding a large planted clique can be solved by an one class SVM.

**Acknowledgements**

CB is grateful to Department of CSE, Chalmers University of Technology for their hospitality and was supported by grants from ICT and Transport Areas of Advance, Chalmers University. VJ and DD were supported by SSF grant *Data Driven Secure Business Intelligence*.

## Footnotes

*Relevant code and datasets can be found on http://www.cse.chalmers.se/~jethava/svm-theta.html

[1] This is equivalent to defining an orthogonal labelling on the Union graph of $G^{(1)}, \ldots, G^{(M)}$

[2] `ftp://dimacs.rutgers.edu/pub/challenge/graph/benchmarks/clique/`

[3] `http://asi.insa-rouen.fr/enseignants/~arakotom/code/mklindex.html`

[4] `http://www.csie.ntu.edu.tw/~cjlin/libsvm/`

# References

[1] Louigi Addario-berry, Nicolas Broutin, Gbor Lugosi, and Luc Devroye. Combinatorial testing problems. *Annals of Statistics*, 38:3063–3092, 2010.

[2] Noga Alon, Michael Krivelevich, and Benny Sudakov. Finding a large hidden clique in a random graph. *Random Structures and Algorithms*, pages 457–466, 1998.

[3] B. Bollobás. *Modern graph theory*, volume 184. Springer Verlag, 1998.

[4] Stephen Boyd and Lieven Vandenberghe. *Convex Optimization*. Cambridge University Press, New York, NY, USA, 2004.

[5] T.-H. Hubert Chan, Kevin L. Chang, and Rajiv Raman. An sdp primal-dual algorithm for approximating the lovász-theta function. In *ISIT*, 2009.

[6] Amin Coja-Oghlan and Anusch Taraz. Exact and approximative algorithms for coloring g(n, p). *Random Struct. Algorithms*, 24(3):259–278, 2004.

[7] U. Feige and D. Ron. Finding hidden cliques in linear time. In *AofA10*, 2010.

[8] Uriel Feige and Robert Krauthgamer. Finding and certifying a large hidden clique in a semi-random graph. *Random Struct. Algorithms*, 16:195–208, March 2000.

[9] Z. Füredi and J. Komlós. The eigenvalues of random symmetric matrices. *Combinatorica*, 1:233–241, 1981.

[10] Michel X. Goemans. Semidefinite programming in combinatorial optimization. *Math. Program.*, 79:143–161, 1997.

[11] J. Håstad. Clique is hard to approximate within $n^{1-\varepsilon}$. *Acta Mathematica*, 182(1):105–142, 1999.

[12] Roger A. Horn and Charles R. Johnson. *Matrix Analysis*. Cambridge University Press, 1990.

[13] Don R. Hush, Patrick Kelly, Clint Scovel, and Ingo Steinwart. Qp algorithms with guaranteed accuracy and run time for support vector machines. *Journal of Machine Learning Research*, 7:733–769, 2006.

[14] D. Jiang and J. Pei. Mining frequent cross-graph quasi-cliques. *ACM Transactions on Knowledge Discovery from Data (TKDD)*, 2(4):16, 2009.

[15] D.S. Johnson and M.A. Trick. *Cliques, coloring, and satisfiability: second DIMACS implementation challenge, October 11-13, 1993*, volume 26. Amer Mathematical Society, 1996.

[16] Donald Knuth. The sandwich theorem. *Electronic Journal of Combinatorics*, 1(A1), 1994.

[17] Michael Krivelevich and Benny Sudakov. Pseudo-random graphs. In *More Sets, Graphs and Numbers*, volume 15 of *Bolyai Society Mathematical Studies*, pages 199–262. Springer Berlin Heidelberg, 2006.

[18] V.E. Lee, N. Ruan, R. Jin, and C. Aggarwal. A survey of algorithms for dense subgraph discovery. *Managing and Mining Graph Data*, pages 303–336, 2010.

[19] L. Lovasz. On the Shannon capacity of a graph. *Information Theory, IEEE Transactions on*, 25(1):1–7, 1979.

[20] C.J. Luz and A. Schrijver. A convex quadratic characterization of the lovász theta number. *SIAM Journal on Discrete Mathematics*, 19(2):382–387, 2006.

[21] Claire Mathieu and Warren Schudy. Correlation clustering with noisy input. In *Proceedings of the Twenty-First Annual ACM-SIAM Symposium on Discrete Algorithms*, SODA '10, pages 712–728, Philadelphia, PA, USA, 2010. Society for Industrial and Applied Mathematics.

[22] P. Pardalos and S. Rebennack. Computational challenges with cliques, quasi-cliques and clique partitions in graphs. *Experimental Algorithms*, pages 13–22, 2010.

[23] V. Spirin and L.A. Mirny. Protein complexes and functional modules in molecular networks. *Proceedings of the National Academy of Sciences*, 100(21):12123, 2003.

[24] Dekel Yael, Gurel-Gurevich Ori, and Peres Yuval. Finding hidden cliques in linear time with high probability. In *ANALCO11*, 2011.

